# Resonance in a Stochastic Neuron Model with Delayed Interaction

**Toru Ohira***
Sony Computer Science Laboratory
3-14-13 Higashi-gotanda
Shinagawa, Tokyo 141, Japan
*ohira@csl.sony.co.jp*

**Yuzuru Sato**
Institute of Physics,
Graduate School of Arts and Science, University of Tokyo
3-8-1 Komaba, Meguro, Tokyo 153 Japan
*ysato@sacral.c.u-tokyo.ac.jp*

**Jack D. Cowan**
Department of Mathematics
University of Chicago
5734 S. University, Chicago, IL 60637, U.S.A
*cowan@math.uchicago.edu*

## Abstract

We study here a simple stochastic single neuron model with delayed self–feedback capable of generating spike trains. Simulations show that its spike trains exhibit resonant behavior between "noise" and "delay". In order to gain insight into this resonance, we simplify the model and study a stochastic binary element whose transition probability depends on its state at a fixed interval in the past. With this simplified model we can analytically compute interspike interval histograms, and show how the resonance between noise and delay arises. The resonance is also observed when such elements are coupled through delayed interaction.

## 1 Introduction

"Noise" and "delay" are two elements which are associated with many natural and artificial systems and have been studied in diverse fields. Neural networks provide representative examples of information processing systems with noise and delay. Though much research has gone into the investigation of these two factors in the community, they have mostly been separately studied (see e.g. [1]). Neural

models incorporating both noise and delay are more realistic [2], but their complex characteristics have yet to be explored both theoretically and numerically.

The main theme of this paper is the study of a simple stochastic neural model with delayed interaction which can generate spike trains. The most striking feature of this model is that it can show a regular spike pattern with suitably "tuned" noise and delay [3]. Stochastic resonance in neural information processing has been investigated by others (see e.g. [4]). This model, however, introduces a different type of such resonance, via delay rather than through an external oscillatory signal. It can be classified with models of stochastic resonance without an external signal [5].

The novelty of this model is the use of delay as the source of its oscillatory dynamics. To gain insight into the resonance, we simplify the model and study a stochastic binary element whose transition probability depends on its state at a fixed interval in the past. With this model, we can analytically compute interspike interval histograms, and show how the resonance between noise and delay arises. We further show that the resonance also occurs when such stochastic binary elements are coupled through delayed interaction.

## 2 Single Delayed-feedback Stochastic Neuron Model

Our model is described by the following equations:

$$\mu \frac{d}{dt} V(t) = -V(t) + W \phi(V(t - \tau)) + \xi_L(t)$$
$$\phi(V(t)) = \frac{2}{1 + e^{-\eta(V(t) - \theta)}} - 1 \tag{1}$$

where $\eta$ and $\theta$ are constants, and $V$ is the membrane potential of the neuron. The noise term $\xi_L$ has the following probability distribution.

$$P(\xi = u) = \frac{1}{2L} \quad (-L \leq u \leq L)$$
$$= 0 \quad (u < -L, u > L), \tag{2}$$

i.e., $\xi_L$ is a time uncorrelated uniformly distributed noise in the range $(-L, L)$. It can be interpreted as a fluctuation that is much faster than the membrane relaxation time $\mu$. The model can be interpreted as a stochastic neuron model with delayed self–feedback of weight $W$, which is an extension of a model with no delay previously studied using the Fokker–Planck equation [6].

We numerically study the following discretized version:

$$V(t + 1) = \frac{2}{1 + e^{-\eta(V(t - \tau) - \theta)}} - 1 + \xi_L \tag{3}$$

We fix $\eta$ and $\theta$ so that this map has two basins of attractors of differing size with no delay, as shown in Figure 1(A). We have simulated the map (3) with various noise widths and delays and find regular spiking behavior as shown in Fig 1(C) for tuned noise width and delay. In case the noise width is too large or too small given self–feedback delay, this rhythmic behavior does not emerge, as shown in Fig1(B) and (D).

We argue that the delay changes the effective shape of the basin of attractors into an oscillatory one, just like that due to an external oscillating force which, as is well-known, leads to stochastic resonance with a tuned noise width. The analysis of the dynamics given by (1) or (3), however, is a non–trivial task, particularly with

respect to the spike trains. A previous analysis using the Fokker–Planck equation cannot capture this emergence of regular spiking behavior. This difficulty motivates us to further simplify our model, as described in the next section.

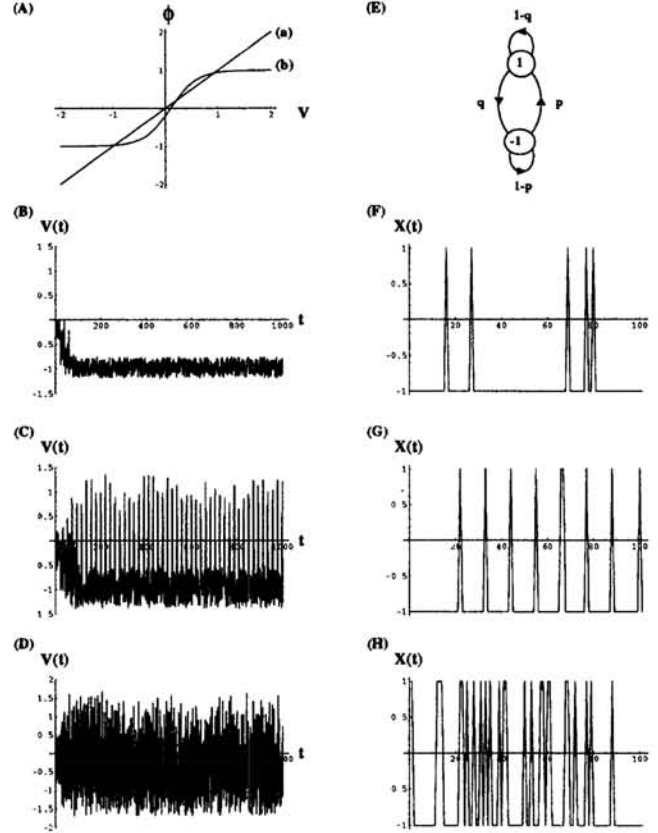

Figure 1: (A) The shape of the sigmoid function $\phi$ (b) for $\eta = 4$ and $\theta = 0.1$. The straight line (a) is $\phi = V$ and the crossings of the two lines indicate the stationary point of the dynamics. Also, the typical dynamics of $V(t)$ from the map model are shown as we change noise width $L$. The values of $L$ are (B) $L = 0.2$, (C) $L = 0.4$, (D) $L = 0.7$. The data is taken with $\tau = 20$, $\eta = 4.0$, $\theta = 0.1$ and the initial condition $V(t) = 0.0$ for $t \in [-\tau, 0]$. The plots are shown between $t = 0$ to 1000. (E) Schematic view of the single binary model. Some typical dynamics from the binary model are also shown. The values of parameters are $\tau = 10$, $q = 0.5$, and (F) $p = 0.005$, (G) $p = 0.05$, and (H) $p = 0.2$.

## 3   Delayed Stochastic Binary Neuron Model

The model we now discuss is an approximation of the dynamics that retains the asymmetric stochastic transition and delay. The state $X(t)$ of the system at time step $t$ is either $-1$ or $1$. With the same noise $\xi_L$, the model is described as follows:

$$
\begin{aligned}
X(t+1) &= \theta[f(X(t-\tau)) + \xi_L], \\
f(n) &= \frac{1}{2}((a+b) + n(a-b)), \\
\theta[n] &= 1 \quad (0 \leq n), \qquad -1 \quad (0 > n),
\end{aligned}
\tag{4}
$$

where $a$ and $b$ are parameters such that $|a| \leq L$ and $|b| \leq L$, and $\tau$ is the delay. This model is an approximate discretization of the space of map (3) into two states

with $a$ and $b$ controlling the bias of transition depending on the state of $X$ $\tau$ steps earlier. When $a \neq b$, the transition between the two states is asymmetric, reflecting the two differing sized basins of attractors.

We can describe this model more concisely in probability space (Figure 1(E)). The formal definition is given as follows:

$$
\begin{aligned}
P(1, t+1) &= p, & X(t-\tau) = -1, \\
&= 1-q, & X(t-\tau) = 1, \\
P(-1, t+1) &= q, & X(t-\tau) = 1, \\
&= 1-p, & X(t-\tau) = -1, \\
p &= \frac{1}{2}(1+\frac{b}{L}), \\
q &= \frac{1}{2}(1-\frac{a}{L}), & (5)
\end{aligned}
$$

where $P(s, t)$ is the probability that $X(t) = s$. Hence, the transition probability of the model depends on its state $\tau$ steps in the past, and is a special case of a delayed random walk [7].

We randomly generate $X(t)$ for the interval $t = (-\tau, 0)$. Simulations are performed in which parameters are varied and $X(t)$ is recorded for up to $10^6$ steps. They appear to be qualitatively similar to those generated by the map dynamics (Figure 1(F),(G),(H)). ¿From the trajectory $X(t)$, we construct a residence time histogram $h(u)$ for the system to be in the state $-1$ for $u$ consecutive steps. Some examples of the histograms are shown in Figure 2 ($q = 1 - q = 0.5$, $\tau = 10$). We note that with $p << 0.5$, as in Figure 2(A), the model has a tendency to switch or spike to the $X = 1$ state after the time step interval of $\tau$. But the spike trains do not last long and result in a small peak in the histogram. For the case of Figure 2(C) where $p$ is closer to 0.5, we observe less regular transitions and the peak height is again small. With appropriate $p$ as in Figure 2(B), spikes tend to appear at the interval $\tau$ more frequently, resulting in higher peaks in the histogram. This is what we mean by stochastic resonance (Figure 2(D)). Choosing an appropriate $p$ is equivalent to "tuning" the noise width $L$, with other parameters appropriately fixed. In this sense, our model exhibits stochastic resonance.

This model can be treated analytically. The first observation to make with the model is that given $\tau$, it consists of statistically independent $\tau + 1$ Markov chains. Each Markov chain has its state appearing at every $\tau+1$ interval. With this property of the model, we label time step $t$ by the two integers $s$ and $k$ as follows

$$
t = s(\tau + 1) + k, \quad (0 \leq s, 0 \leq k \leq \tau) \tag{6}
$$

Let $P_{\pm}(t) \equiv P_{\pm}(s, k)$ be the probability for the state to be in the $\pm 1$ state at time $t$ or $(s, k)$. Then, it can be shown that

$$
\begin{aligned}
P_+(s, k) &= \alpha(1 - \gamma^s) + \gamma^s P_+(s = 0, k), \\
P_-(s, k) &= \beta(1 - \gamma^s) + \gamma^s P_-(s = 0, k), \\
\alpha &= \frac{p}{p+q}, \\
\beta &= \frac{q}{p+q}, \\
\gamma &= 1 - (p+q). \tag{7}
\end{aligned}
$$

In the steady state, $P_+(s \to \infty, k) \equiv P_+ = \alpha$ and $P_-(s \to \infty, k) \equiv P_- = \beta$. The steady state residence time histogram can be obtained by computing the following

quantity, $h(u) \equiv P(+; -, u; +)$, which is the probability that the system takes consecutive $-1$ states $u$ times between two $+1$ states. With the definition of the model and statistical independence between Markov chains in the sequence, the following expression can be derived:

$$P(+; -, u; +) = P_+(P_-)^u P_+ = (\beta)^u (\alpha)^2 \quad (1 \le u < \tau) \tag{8}$$

$$= P_+(P_-)^\tau (1 - q) = (\beta)^\tau (\alpha)(1 - q) \quad (u = \tau) \tag{9}$$

$$= P_+(P_-)^\tau (q)(1 - p)^{u - \tau}(p) = (\beta)^u (p)^2 \quad (u > \tau) \tag{10}$$

With appropriate normalization, this expression reflects the shape of the histogram obtained by numerical simulations, as shown in Figure 2. Also, by differentiating equation (9) with respect to $p$, we derive the resonant condition for the peak to reach maximum height as

$$q = p\tau \tag{11}$$

or, equivalently,

$$L - a = (L + b)\tau. \tag{12}$$

In Figure 2(D), we see that maximum peak amplitude is reached by choosing parameters according to equation (11). We note that this analysis for the histogram is exact in the stationary limit, which makes this model unique among those showing stochastic resonance.

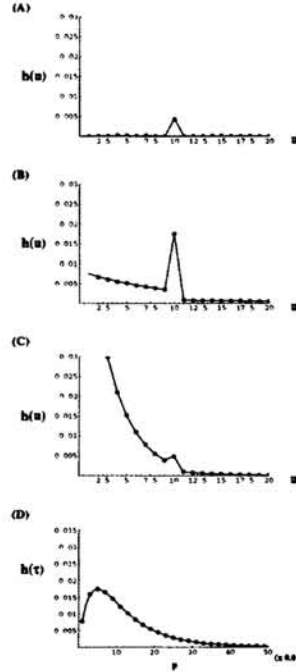

Figure 2: Residence time histogram and dynamics of $X(t)$ as we change $p$. The values of $p$ are (A) $p = 0.005$, (B) $p = 0.05$, (C) $p = 0.2$. The solid line in the histogram is from the analytical expression given in equations (8-10). Also, in (D) we show a plot of peak height by varying $p$. The solid line is from equation (9). The parameters are $\tau = 10$, $q = 0.5$.

## 4   Delay Coupled Two Neuron Case

We now consider a circuit comprising two such stochastic binary neurons coupled with delayed interaction. We observe again that resonance between noise and delay

takes place. The coupled two neuron model is a simple extension of the model in the previous section. The transition probability of each neuron is dependent on the other neuron's state at a fixed interval in the past. Formally, it can be described in probability space as follows.

$$
\begin{aligned}
P_1(1, t+1) &= p_1, & X_2(t-\tau_2) &= -1, \\
&= 1-q_1, & X_2(t-\tau_2) &= 1, \\
P_1(-1, t+1) &= q_1, & X_2(t-\tau_2) &= 1, \\
&= 1-p_1, & X_2(t-\tau_2) &= -1, \\
P_2(1, t+1) &= p_2, & X_1(t-\tau_1) &= -1, \\
&= 1-q_2, & X_1(t-\tau_1) &= 1, \\
P_2(-1, t+1) &= q_2, & X_1(t-\tau_1) &= 1, \\
&= 1-p_2, & X_1(t-\tau_1) &= -1
\end{aligned} \tag{13}
$$

$P_i(s, t)$ is the probability that the state of the neuron $i$ is $X_i(t) = s$. We have performed simulation experiments on the model and have again found resonance between noise and delay. Though more intricate than the single neuron model, we can perform a similar theoretical analysis of the histograms and have obtained approximate results for some cases. For example, we obtain the following approximate analytical results for the peak height of the interspike histogram of $X_1$ for the case $\tau_1 = \tau_2 \equiv \tau$. ( The peak occurs at $\tau_1 + \tau_2 + 1$.)

$$
\begin{aligned}
H(p_1, p_2, q_1, q_2) &= \{\mu_3(p_1, p_2, q_1, q_2)q_1 + \mu_4(p_1, p_2, q_1, q_2)(1-p_1)\}^\tau \\
&\quad \{\mu_1(p_1, p_2, q_1, q_2)(q_1 q_2 p_1 + q_1(1-q_2)(1-q_1)) \\
&\quad + \mu_2(p_1, p_2, q_1, q_2)((1-p_1)q_2 p_1 + (1-p_1)(1-q_2)(1-q_1))\}
\end{aligned} \tag{14}
$$

$$
\mu_1(p_1, p_2, q_1, q_2) = \frac{f_1(p_1, p_2, q_1, q_2) f_2(p_1, p_2, q_1, q_2)}{s(p_1, p_2, q_1, q_2)} \tag{15}
$$

$$
\mu_2(p_1, p_2, q_1, q_2) = \frac{f_1(p_1, p_2, q_1, q_2)}{s(p_1, p_2, q_1, q_2)} \tag{16}
$$

$$
\mu_3(p_1, p_2, q_1, q_2) = \frac{f_2(p_1, p_2, q_1, q_2)}{s(p_1, p_2, q_1, q_2)} \tag{17}
$$

$$
\mu_4(p_1, p_2, q_1, q_2) = \frac{1}{s(p_1, p_2, q_1, q_2)} \tag{18}
$$

$$
f_1(p_1, p_2, q_1, q_2) = \frac{p_1(1-p_2) + p_2(1-q_1)}{q_1(1-q_2) + q_2(1-q_1)} \tag{19}
$$

$$
f_2(p_1, p_2, q_1, q_2) = \frac{p_2 + p_1(1-p_2-q_2)}{q_2 + q_1(1-p_2-q_2)} \tag{20}
$$

$$
\begin{aligned}
s(p_1, p_2, q_1, q_2) &= f_1(p_1, p_2, q_1, q_2) f_2(p_1, p_2, q_1, q_2) \\
&\quad + f_1(p_1, p_2, q_1, q_2) + f_2(p_1, p_2, q_1, q_2) + 1
\end{aligned} \tag{21}
$$

These analytical results are compared with the simulation experiments, examples of which are shown in Figure 3. A detailed analysis, particularly for the case of $\tau_1 \neq \tau_2$, is quite intricate and is left for the future.

## 5  Discussion

There are two points to be noted. Firstly, although there are examples which may indicate that stochastic resonance is utilized in biological information processing, it is yet to be explored if the resonance between noise and delay has some role in

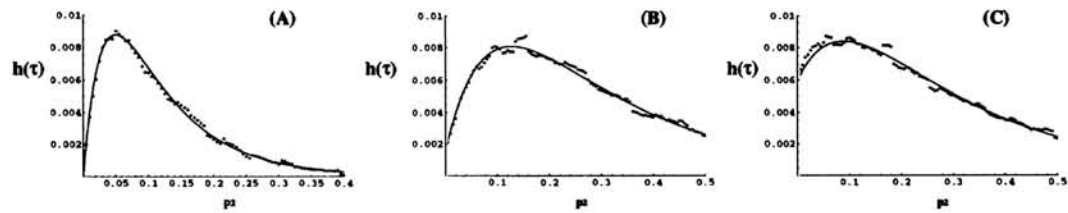

Figure 3: A plot of peak height by varying $p_2$. The solid line is from equation (14-20). The parameters are $\tau_1 = \tau_2 = 10$, $q_1 = q_2 = 0.5$, (A)$p_1 = p_2$, (B) $p_1 = 0.005$, (C) $p_1 = 0.025$.

neural information processing. Secondly, there are many investigations of spiking neural models and their applications (see e.g., [8]). Our model can be considered as a new mechanism for generating controlled stochastic spike trains. One can predict its application to weak signal transmission analogous to recent research using stochastic resonance with a larger number of units in series [9]. Investigations of the network model with delayed interactions are currently underway.

## Footnotes

*Affiliated also with the Laboratory for Information Synthesis, RIKEN Brain Science Institute, Wako, Saitama, Japan

## References

[1] Hertz, J. A., Krogh, A., & Palmer, R. G. (1991). *Introduction to the Theory of Neural Computation*. Redwood City: Addison–Wesley.

[2] Foss, J., Longtin, A., Mensour, B., & Milton, J. G. (1996). Multistability and Delayed Recurrent Loops. *Physical Review Letters, 76*, 708–711; Pham, J., Pakdaman, K., Vibert, J.-F. (1998). Noise–induced coherent oscillations in randomly connected neural networks. *Physical Review E, 58*, 3610–3622; Kim, S., Park, S. H., Pyo, H.-B. (1999). Stochastic Resonance in Coupled Oscillator Systems with Time Delay. *Physical Review Letters, 82*, 1620–1623; Bressloff, P. C. (1999). Synaptically Generated Wave Propagation in Excitable Neural Media. *Physical Review Letters, 82*, 2979–2982.

[3] Ohira, T. & Sato, Y. (1999). Resonance with Noise and Delay. *Physical Review Letters, 82*, 2811–2815.

[4] Gammaitoni, L., Hänggi, P., Jung, P., & Marchesoni, F.(1998). Stochastic Resonance. *Review of Modern Physics, 70*, 223–287.

[5] Gang, H., Ditzinger, T., Ning, C. Z., & Haken, H.(1993) Stochastic Resonance without External Periodic Force. *Physical Review Letters, 71*, 807–810; Rappel, W-J. & Strogatz, S. H. (1994). Stochastic resonance in an autonomous system with a nonuniform limit cycle. *Physical Review E, 50*, 3249–3250; Longtin, A. (1997). Autonomous stochastic resonance in bursting neurons. *Physical Review E, 55*, 868–876.

[6] Ohira, T. & Cowan J. D. (1995). Stochastic Single Neurons, *Neural Communication, 7* 518–528.

[7] Ohira, T. & Milton, J. G. (1995). Delayed Random Walks. *Physical Review E, 52*, 3277–3280; Ohira, T. (1997). Oscillatory Correlation of Delayed Random Walks, *Physical Review E, 55*, R1255–1258.

[8] Maas, W. (1997). Fast Sigmoidal Network via Spiking Neurons. *Neural Computation, 9*(2), 279–304; Maas, W. (1996). Lower Bounds for the Computational Power of Networks of Spiking Neurons. *Neural Computation, 8*(1), 1–40.

[9] Löcher, M., Cigna, D., and Hunt, E. R. (1998). Noise Sustained Propagation of a Signal in Coupled Bistable Electric Elements *Physical Review Letters, 80*, 5212–5215.
